# Lifted Inference Seen from the Other Side : The Tractable Features

**Abhay Jha   Vibhav Gogate   Alexandra Meliou   Dan Suciu**
Computer Science & Engineering
University of Washington
Washington, WA 98195
{abhaykj,vgogate,ameli,suciu}@cs.washington.edu

## Abstract

Lifted Inference algorithms for representations that combine first-order logic and graphical models have been the focus of much recent research. All lifted algorithms developed to date are based on the same underlying idea: take a standard probabilistic inference algorithm (e.g., variable elimination, belief propagation etc.) and improve its efficiency by exploiting repeated structure in the first-order model. In this paper, we propose an approach from the other side in that we use techniques from logic for probabilistic inference. In particular, we define a set of rules that look only at the logical representation to identify models for which exact efficient inference is possible. Our rules yield new tractable classes that could not be solved efficiently by any of the existing techniques.

## 1   Introduction

Recently, there has been a push towards combining logical and probabilistic approaches in Artificial Intelligence. It is motivated in large part by the representation and reasoning challenges in real world applications: many domains such as natural language processing, entity resolution, target tracking and Bio-informatics contain both rich relational structure, and uncertain and incomplete information. Logic is good at handling the former but lacks the representation power to model the latter. On the other hand, probability theory is good at modeling uncertainty but inadequate at handling relational structure.

Many representations that combine logic and graphical models, a popular probabilistic representation [1, 2], have been proposed over the last few years. Among them, Markov logic networks (MLNs) [2, 3] are arguably the most popular one. In its simplest form, an MLN is a set of weighted first-order logic formulas, and can be viewed as a template for generating a Markov network. Specifically, given a set of constants that model objects in the domain, it represents a ground Markov network that has one (propositional) feature for each grounding of each (first-order) formula with constants in the domain.

Until recently, most inference schemes for MLNs were propositional: inference was carried out by first constructing a ground Markov network and then running a standard probabilistic inference algorithm over it. Unfortunately, the ground Markov network is typically quite large, containing millions and sometimes even billions of inter-related variables. This precludes the use of existing probabilistic inference algorithms, as they are unable to handle networks at this scale. Fortunately, in some cases, one can perform *lifted inference* in MLNs without grounding out the domain. Lifted inference treats sets of indistinguishable objects as one, and can yield exponential speed-ups over propositional inference.

Many lifted inference algorithms have been proposed over the last few years (c.f. [4, 5, 6, 7]). All of them are based on the same principle: take an existing probabilistic inference algorithm and try

| Interpretation in English | Feature | Weight |
|---|---|---|
| Most people don't smoke | `¬Smokes(X)` | 1.4 |
| Most people don't have asthma | `¬Asthma(X)` | 2.3 |
| Most people aren't friends | `¬Friends(X,Y)` | 4.6 |
| People who have asthma don't smoke | `Asthma(X) ⇒ ¬Smokes(X)` | 1.5 |
| Asthmatics don't have smoker friends | `Asthma(X) ∧ Friends(X,Y) ⇒ ¬Smokes(Y)` | 1.1 |

Table 1: An example MLN (modified from [10]).

to lift it by carrying out inference over groups of random variables that behave similarly during the algorithm's execution. In other words, these algorithms are basically lifted versions of standard probabilistic inference algorithms. For example, first-order variable elimination [4, 5, 7] lifts the standard variable elimination algorithm [8, 9], while lifted Belief propagation [10] lifts Pearl's Belief propagation [11, 12].

In this paper, we depart from existing approaches, and present a new approach to lifted inference from the other, logical side. In particular, we propose a set of rewriting rules that exploit the structure of the logical formulas for inference. Each rule takes an MLN as input and expresses its partition function as a combination of partition functions of *simpler* MLNs (if the preconditions of the rule are satisfied). Inference is tractable if we can evaluate an MLN using these set of rules. We analyze the time complexity of our algorithm and identify new tractable classes of MLNs, which have not been previously identified.

Our work derives heavily from database literature in which inference techniques based on manipulating logical formulas (queries) have been investigated rigorously [13, 14]. However, the techniques that they propose are not lifted. Our algorithm extends their techniques to lifted inference, and thus can be applied to a strictly larger class of probabilistic models.

To summarize, our algorithm is truly lifted, namely we never ground the model, and it offers guarantees on the running time. This comes at a cost that we do not allow arbitrary MLNs. However, the set of tractable MLNs is quite large, and includes MLNs that cannot be solved in PTIME by any of the existing lifted approaches. The small toy MLN given in Table 1 is one such example. This MLN is also out of reach of state-of-the-art propositional inference approaches such as variable elimination [8, 9], which are exponential in treewidth. This is because the treewidth of the ground Markov network is polynomial in the number of constants in the domain.

## 2   Preliminaries

In this section we will cover some preliminaries and notation used in the rest of the paper. A *feature* ($f_i$) is constructed using *constants*, *variables*, and *predicates*. Constants, denoted with small-case letters (e.g. $a$), are used to represent a particular object. An upper-case letter (e.g. $X$) indicates a variable associated with a particular domain ($\Delta_X$), ranging over all objects in its domain. Predicate symbols (e.g. `Friends`) are used to represent relationships between the objects. For example, `Friends(bob,alice)` denotes that Alice (represented by constant `alice`) and Bob (constant `bob`) are friends. An *atom* is a predicate symbol applied to a tuple of *variables* or *constants*. For example, `Friends(bob,X)` and `Friends(bob,alice)` are atoms.

A *conjunctive feature* is of the form $\forall \bar{X} \; r_1 \wedge r_2 \wedge \cdots \wedge r_k$, where each $r_i$ is an atom or the negation of an atom, and $\bar{X}$ are the variables used in the atoms. Similarly, a *disjunctive feature* is of the form $\forall \bar{X} \; r_1 \vee r_2 \vee \cdots \vee r_k$. For example, $f_c : \forall X \; ¬\text{Smokes(X)} \wedge \text{Asthma(X)}$ is a conjunctive feature, while $f_d : \forall X \; ¬\text{Smokes(X)} \vee ¬\text{Friends(bob,X)}$ is a disjunctive feature. The former asserts everyone in the domain of $X$ has asthma and does not smoke. The latter says that if a person smokes, he/she cannot be friends with Bob. A *grounding* of a feature is an assignment of the variables to constants from their domain. For example, `¬Smokes(alice) ∨ ¬Friends(bob,alice)` is a grounding of the disjunctive feature $f_d$. We assume that no predicate symbol occurs more than once in a feature i.e. we don't allow for self-joins. In this work we focus on features containing only universal quantifiers ($\forall$), and will from now on drop the quantification symbol $\forall$ from the notation.

Given a set $(w_i, f_i)_{i=1,k}$ where each $f_i$ is a conjunctive or disjunctive feature and $w_i \in \mathbb{R}$ is a weight assigned to that feature, we define the following probability distribution over a possible

world $\omega$ in accordance with *Markov Logic Networks* (MLN) :

$$Pr(\omega) = \frac{1}{Z} \exp\left(\sum_i w_i N(f_i, \omega)\right) \qquad (1)$$

In Equation (1), a possible world $\omega$ can be any subset of tuples from the domain of predicates, $Z$, the normalizing constant is called the partition function, and $N(f_i, \omega)$ is the number of groundings of feature $f_i$ that are true in the world $\omega$.

Table 1 gives an example of a MLN that has been modified from [10]. There is an implicit type-safety assumption in the MLNs, that if a predicate symbol occurs in more than one feature, then the variables used at the same position must have same domain. In the MLN of Table 1, if $\Delta_X = \Delta_Y = \{alice, bob\}$; then predicates Smokes and Asthma each have two tuples, while Friends has four. Hence, the total number of possible worlds is $2^{2+2+4} = 256$. Consider the possible world $\omega$ below :

| Smokes | Asthma | Friends |
|--------|--------|---------|
| bob    | bob    | (bob,bob) |
|        |        | (bob,alice) |
|        | alice  |         |
|        |        | (alice,alice) |

Then from Equation (1): $Pr(\omega) = \frac{1}{Z} \exp\left(1.4 \cdot 1 + 2.3 \cdot 0 + 4.6 \cdot 0 + 1.5 \cdot 1 + 1.1 \cdot 2\right)$. In this paper we focus on MLNs, but our algorithm is applicable to other first order probabilistic models as well.

## 3  Problem Statement

In this paper, we are interested in computing the partition function $Z(M)$ of an MLN $M$. We formulate the partition function in a parametrized form, using the notion of Generating Functions of *Counting Programs* (CP). A Counting Program is a set of features $\bar{f}$ along with indeterminates $\bar{\alpha}$, where $\alpha_i$ is the indeterminate for $f_i$. Given a counting program $P = (f_i, \alpha_i)_{i=1...k}$, we define its *generating function*(GF) $F_P$ as follows:

$$F_P(\bar{\alpha}) = \sum_\omega \prod_i \alpha_i^{N(f_i, \omega)} \qquad (2)$$

The generating function as expressed in Eq. 2 is in general of exponential size in the domain of objects. We want to characterize cases where we can express it more succinctly, and hence compute the partition function faster. Let $n$ be the size of the object domain, and $k$ be the size of our program. Then we are interested in the cases where $F_P$ can be computed with following number of arithmetic operations.

**Closed Form** Polynomial in $\log(n), k$
**Polynomial Expression** Polynomial in $n, k$
**Pseudo-Polynomial Expression** Polynomial in $n$ for bounded $k$

Computing $F_P$ refers to evaluating it for one instantiation of parameters $\bar{\alpha}$. To illustrate the above cases, let $k = 1$. Then the pseudo-polynomial and polynomial expression are equivalent. The program $(R(X, Y), \alpha)$ has GF $(1 + \alpha)^{|\Delta_X||\Delta_Y|}$, which is in closed form. While the program $(R(X) \wedge S(X, Y) \wedge T(Y), \alpha)$ has GF $2^{|\Delta_X||\Delta_Y|} \sum_{i=0}^{|\Delta_X|} \binom{|\Delta_X|}{i} \left(1 + \left(\frac{1+\alpha}{2}\right)^i\right)^{|\Delta_Y|}$, which is a polynomial expression. This polynomial does not have a closed form.

In the following section we demonstrate an algorithm that computes the generating function, and allows us to identify cases where the generating function falls under one of the above categories.

## 4  Computing the Generating Function

Asssume a Counting Program $P = (f_i, \alpha_i)_{i=1,k}$. In this section, we present some rules that can be used to compute the GF of a CP from simpler CPs. We can then upper bound the size of $F_P$ by the

choice of rules used. The cases which cannot be evaluated by these rules are still open and we don't know if the GF in those cases can be expressed succinctly.

We will require that all CPs are in *normal form* to simplify our analysis. Note that the normality requirement does not change the class of CPs that can be solved in PTIME by our algorithm. This is because every CP can converted to an equivalent normal CP in PTIME.

## 4.1 Normal Counting Programs

**Definition 4.1** *A counting program is called **normal** if it satisfies the following properties :*
   1. *There are no constants in any feature.*
   2. *If two distinct atoms with the same predicate symbol have variables $X$ and $Y$ in the same position, then $\Delta_X = \Delta_Y$.*

It is easy to show that:

**Proposition 4.2** *Computing the partition function of an MLN can be reduced in PTIME to computing the generating function of a normal CP.*

The following example demonstrates how to normalize a set of features.

**Example 4.3** Consider a CP containing two features `Friends`$(X, Y)$ and `Friends`$(bob, Y)$. Clearly, it is not in normal form because the second feature contains a constant. To normalize it, we can replace the two features by: (i) `Friends1`$(Y) \equiv$ `Friends`$(bob, Y)$, and (ii) `Friends2`$(Z, Y) \equiv$ `Friends`$(X, Y), X \neq bob$, where the domain of $Z$ is $\Delta_Z = \Delta_X \setminus bob$.

Note that we assume criterion 2 is satisfied in MLNs. During the course of algorithm, it may get violated when we replace variables with constants as we'll see, but we can use the above transformation whenever that happens. So from now on we assume that our CP is normalized.

## 4.2 Preliminaries and Operators

We proceed to establish notation and operators used by our algorithm. Given a feature $f$, we denote by $Vars(f)$ the set of variables used in its atoms. We assume that variables used in different features must be different. Furthermore, without loss of generality, we assume numeric domains for each logical variable, namely $\Delta_X = \{1, \ldots, |\Delta_X|\}$. We define a **substitution** $f[a/X]$, where $X \in Vars(f)$ and $a \in \Delta_X$, as the replacement of $X$ with $a$ in every atom of $f$. $P[a/X]$ applies the substitution $f_i[a/X]$ to every feature $f_i$ in $P$. Note that after a substitution, the CP is no longer normal and therefore, we may have to normalize it.

Define a relation $U$ among the variables of a CP as follows : $U(X, Y)$ iff there exist two atoms $r_i, r_j$ with the same predicate, such that $X \in Vars(r_i), Y \in Vars(r_j)$, and $X$ and $Y$ appear at the same position in $r_i$ and $r_j$ respectively. Let $\mathcal{U}$ be the transitive closure of $U$. Note that $\mathcal{U}$ is an equivalence relation. For a variable $X$, denote by $Unify(X)$ its equivalence class under $\mathcal{U}$. For example, given two features `Smokes(X)` $\wedge$ `¬Asthma(X)` and `¬Smokes(Y)` $\vee$ `¬Friends(Z,Y)`, we have $Unify(X) = Unify(Y) = \{X, Y\}$. Given a feature, a variable is a **root variable** iff it appears in every atom of the feature. For some variable $X$, the set $\mathcal{X} = Unify(X)$ is a **separator** if $\forall Y \in \mathcal{X}$ : $Y \in Vars(f_i)$ implies $Y$ must be a root variable for $f_i$. In the last example, the set $\{X, Y\}$ is a separator. Notice that, since the program is normal, we have $\Delta_X = \Delta_Y$ whenever $Y \in Unify(X)$, thus, if $\bar{X}$ is a separator, then we write $\Delta_{\bar{X}}$ for $\Delta_Y$ for any $Y \in Unify(X)$. Two variables are called **equivalent** if there is a bijection from $Unify(X)$ to $Unify(Y)$ such that for any $Z_1 \in Unify(X)$ and its image $Z_2 \in Unify(Y)$, $Z_1$ and $Z_2$ always occur together.

Next, we define three operators used by our algorithm: splitting, conditioning and Dirichlet convolution. We define a process $Split(Y, k)$ that splits every feature in the CP that contains the variable $Y$ into two features with disjoint domains: one with $\Delta_Y = \{k\}$ and the other with $\Delta_{Y^c} = \Delta_Y - \{k\}$. Both features retain the same indeterminate. Also, $Cond(i, r, k)$ defines a process that removes an atom $r$ from feature $f_i$. Denote $f_i' = f_i \setminus \{r\}$; then $Cond(i, r, k)$ replaces $f_i$ with (i) two features $(TRUE, \alpha_i^k)$ and $(f_i', 1)$ if $r \Rightarrow f_i$, (ii) one feature $(f_i', 1)$ if $r \Rightarrow \neg f_i$, and (iii) $(f_i', \alpha_i)$ otherwise.

Given two polynomials $P = \sum_i^n a_i \alpha^i$ and $Q = \sum_i^m b_i \alpha^i$, their **Dirichlet convolution**, $P * Q$, is defined as:

$$P * Q = \sum_{i,j} a_i b_j \alpha^{ij}$$

We define a new variant of this operator $P *^c Q$ as: $P *^c Q = \alpha^{mn} P'\left(\frac{1}{\alpha}\right) * Q'\left(\frac{1}{\alpha}\right)$, where $P'\left(\frac{1}{\alpha}\right) = \frac{P(\alpha)}{\alpha^n}$ and $Q'\left(\frac{1}{\alpha}\right) = \frac{Q(\alpha)}{\alpha^m}$

## 4.3 The Algorithm

Our algorithm is basically a recursive application of a series of rewriting rules (see rules **R1**-**R6** given below). Each (non-trivial) rule takes a CP as input and if the preconditions for applying it are satisfied, then it expresses the generating function of the input CP as a combination of generating functions of a few simpler CPs. The generating function of the resulting CPs can then be computed (independently) by recursively calling the algorithm on each. The recursion terminates when the generating function of the CP is trivial to compute (SUCCESS) or when none of the rules can be applied (FAILURE). In the case, when algorithm succeeds, we analyze whether the GF is in closed form or is a polynomial expression.

Next, we present our algorithm which is essentially a sequence of rules. Given a CP, we go through the rules in order and apply the first applicable rule, which may require us to recursively compute the GF of simpler CPs, for which we continue in the same way.

Our first rule uses feature and variable equivalence to reduce the size of the CP. Formally,

**Rule R1 (Variable and Feature Equivalence Rule)** *If variables $X$ and $Y$ are equivalent, replace the pair with a single new variable $Z$ in every atom where they occur. Do the same for every pair of variables in $Unify(X), Unify(Y)$.*

*If two features $f_i, f_j$ are identical, then we replace them with a single feature $f_i$ with indeterminate $\alpha_i \alpha_j$ that is the product of their individual indeterminates.*

The correctness of **Rule R1** is immediate from the fact that the CP after the transformation is equal to the CP before the transformation.

Our second rule specifies some trivial manipulations.

**Rule R2 (Trivial manipulations)**

1. *Eliminate FALSE features.*
2. *If a feature $f_i$ is $TRUE$, then $F_P = \alpha_i F_{P-f_i}$.*
3. *If a program $P$ is just a tuple then $F_P = 1 + \alpha$, where $\alpha$ is the indeterminate.*
4. *If some feature $f_i$ has indeterminate $\alpha_i = 1$ (due to **R6**), then remove all the atoms in $f_i$ of a predicate symbol that is present in some other feature. Let $N$ be the product of the domain of the rest of the atoms, then $F_P = 2^N F_{P-f_i}$.*

Our third rule utilizes the *independence property*. Intuitively, given two CPs which are independent, namely they have no atoms in common, the generating function of the joint CP is simply the product of the generating function of the two CPs. Formally,

**Rule R3 (Independence Rule)** *If a CP $P$ can be split into two programs $P_1$ and $P_2$ such that the two programs don't have any predicate symbols in common, then $F_P = F_{P_1} \cdot F_{P_2}$.*

The correctness of **Rule R3** follows from the fact that every world $\omega$ of $P$ can be written as a concatenation of two disjoint worlds, namely $\omega = (\omega_1 \cup \omega_2)$ where $\omega_1$ and $\omega_2$ are the worlds from $P_1$ and $P_2$ respectively. Hence the GF can be written as:

$$F_P = \sum_{\omega_1 \cup \omega_2} \prod_{f_i \in P_1} \alpha_i^{N(f_i, \omega_1)} \prod_{f_i \in P_2} \alpha_i^{N(f_i, \omega_2)} = \sum_{\omega_1} \prod_{f_i \in P_1} \alpha_i^{N(f_i, \omega_1)} \sum_{\omega_2} \prod_{f_i \in P_2} \alpha_i^{N(f_i, \omega_2)} = F_{P_1} \cdot F_{P_2} \quad (3)$$

The next rule allows us to split a feature if it has a component that is independent of the rest of the program. Note that while the previous rule splits the program into two independent sets of features, this feature enables us to split a single feature.

**Rule R4 (Dirichlet Convolution Rule)** *If the program contains feature $f = f_1 \wedge f_2$, s.t. $f_1$ doesn't share any variables or symbols with any atom in the program, then $F_P = F_{f_1} * F_{P-f+f_2}$. Similarly if $f = f_1 \vee f_2$, then $F_P = F_{f_1} *^c F_{P-f+f_2}$.*

We show the proof for a single feature $f$, the extension is straightforward. For this, we write GF in a different form as

$$F_P(\alpha) = \sum_i C(f, i)\alpha^i$$

where the coefficient $C(f, i)$ is exactly the number of worlds where the feature $f$ is satisfied $i$ times. Now assume $f = f_1 \wedge f_2$, then in any given world $\omega$, if $f_1$ is satisfied $n_1$ times and $f_2$ is satisfied $n_2$ times, then $f$ is satisfied $n_1 n_2$ times. Hence

$$F_f(\alpha) = \sum_i C(f, i)\alpha^i = \sum_{i_1, i_2 | i_1 i_2 = i} C(f_1, i_1) C(f_2, i_2)\alpha^i = F_{f_1} * F_{f_2}$$

Our next rule utilizes the similarity property in addition to the independence property. Given a set **P** of independent but equivalent CPs, the generating function of the joint CP equals the generating function of any CP, $P_i \in$ **P** raised to the power $|\mathbf{P}|$. By definition, every instantiation $\bar{a}$ of a separator $\bar{X}$ defines a CP that has no tuple in common with other programs for $\bar{X} = \bar{b}$, $\bar{a} \neq \bar{b}$. Moreover, all such CPs are equivalent (subject to a renaming of the variables and constants). Thus, we have the following rule:

**Rule R5 (Power Rule)** *Let $\bar{X}$ be a separator. Then $F_P = \left(F_{P[\bar{a}/\bar{X}]}\right)^{|\Delta_{\bar{x}}|}$*

**Rule R5** generalizes the inversion and partial inversion operators given in [4, 5]. Its correctness follows in a straight-forward manner from the correctness of the independence rule.

Our final rule generalizes the counting arguments presented in [5, 7]. Consider a singleton atom $R(X)$. Conditioning over all possible truth assignments to all groundings of $R(X)$ will yield $2^{|\Delta_x|}$ independent CPs. Thus, the GF can be written as a sum over the generating functions of $2^{|\Delta_x|}$ independent CPs. However, the resulting GF has exponential complexity. In some cases, however, the sum can be written efficiently by grouping together GFs that are equivalent.

**Rule R6 (Generalized Binomial Rule)** *Let $Pred(X)$ be a singleton atom in some feature. For every $Y \in Unify(X)$ apply $Split(Y, k)$. Then for every feature $f_i$ in the new program containing an atom $r = Pred(Y)$ apply $(f_i, \alpha_i) \leftarrow Cond(i, r, k)$ and similarly $(f_i, \alpha_i) \leftarrow Cond(i, \neg r, \Delta_{Y^c} - k)$ for those containing $r = Pred(Y^c)$. Let the resulting program be $P_k$. Then $F_P = \sum_{k=0}^{\Delta_x} \binom{\Delta_x}{k} F_{P_k}$. Note that $P_k$ is just one CP whose GF has a parameter $k$.*

The proof is a little involved and omitted here for lack of space.

Having specified the rules and established their correctness, we now present the main result of this paper:

**Theorem 4.4** *Let $P$ be a Counting Program (CP).*
- *If $P$ can be evaluated using only rules **R1**, **R2**, **R3** and **R5**, then it has a **closed form**.*
- *If $P$ can be evaluated using only rules **R1**, **R2**, **R3**, **R4**, and **R5**, then it has a **polynomial expression**.*
- *If $P$ can be evaluated using rules **Rules 1 to 6** then it admits a **pseudo-polynomial expression.***

Computing the dirichlet convolution (**Rule R4**) requires going through all the coefficients, hence it takes linear time. Thus, we do not have a closed form solution when we apply (**Rule R4**). **Rule R6** implies that we have to recurse over more than one program, hence their repeated application can mean we have to solve number of programs that is exponential in the size of program. Therefore, we can only guarantee a pseudo-polynomial expression if we use this rule.

We can now see the effectiveness of generating functions. When we want to recurse over a set of features, simply keeping the partition function for smaller features is not enough; we need more information than that. In particular we need all the coefficients of the generating function. For e.g. we can't compute the partition function for $R(X) \wedge S(Y)$ with just the partition functions of $R(X)$ and $S(Y)$. However, if we have their GF, the GF of $f = R(X) \wedge S(Y)$ is just a dirichlet convolution of the GF of $R(X)$ and $S(Y)$. One could also compute the GF of $f$ using a dynamic programming algorithm, which keeps all the coefficients of the generating function. Generating functions let us store this information in a very succinct way. For e.g. if the GF is $(1 + \alpha)^n$, then it is much simpler to use this representation, than keeping all $n + 1$ binomial coefficients : $\binom{n}{k}, k = 0, n$.

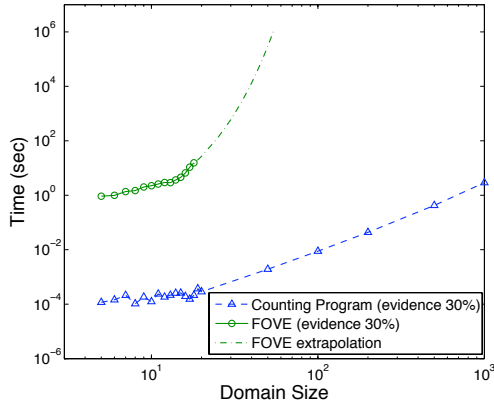
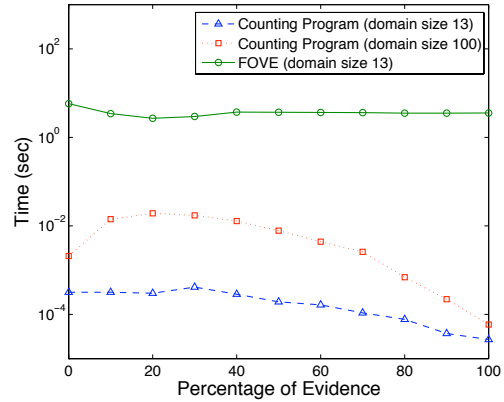

Figure 1: Our approach vs FOVE for increasing domain sizes. X,Y-axes drawn on a log-scale.

Figure 2: Our approach vs FOVE as the evidence increases. Y-axis is drawn on a log scale.

### 4.4 Examples

We illustrate our approach through examples. We will use simple predicate symbols like $R, S, T$ and assume the domain of all variables as [n]. Note that for a single tuple, say $R(a)$ with indeterminate $\alpha$, GF = $1 + \alpha$ from rule **R2**. Now suppose we have a simple program like $P = \{(R(X), \alpha)\}$ (a single feature $R(X)$ with indeterminate $\alpha$). Then from rule **R5**: $F_P = \left(F_{P[a/X]}\right)^n = (1 + \alpha)^n$. These are both examples of programs with *closed form* GF. We can evaluate $F_P$ with $O(\log(n))$ arithmetic operations, while if we were to write the same GF as $\sum_k \binom{n}{k} \alpha^k$ it would require $O(n \log(n))$ operations. The key insight of our approach is representing GFs succinctly. Now assume the following program $P$ with multiple features :

$$
\begin{aligned}
R(X_1) \wedge S(X_1, Y_1) & \quad \alpha \\
S(X_2, Y_2) \wedge T(X_2) & \quad \beta
\end{aligned}
$$

Note that $(X_1, X_2)$ form a separator. Hence using **R5**, $F_P = \left(F_{P[(a,a)/(X_1, X_2)]}\right)^n$. Now consider program $P' = P[(a,a)/(X_1, X_2)]$:

$$
\begin{aligned}
R(a) \wedge S(a, Y_1) & \quad \alpha \\
S(a, Y_2) \wedge T(a) & \quad \beta
\end{aligned}
$$

Using **R4** twice, for $R(a)$ and $T(a)$ along with **R2** (to get the GF for $R(a), T(a)$); we get $F_{P'} = (1 + \alpha)*(1 + \beta)*F_{P''}$, where $P''$ is

$$
\begin{aligned}
S(a, Y_1) & \quad \alpha \\
S(a, Y_2) & \quad \beta
\end{aligned}
$$

which is same as $(S(a, Y), \alpha\beta)$ using **R1**. The GF for this program, as shown earlier is $(1 + \alpha\beta)^n$. Now putting values back together, we get:

$$
F_{P'} = (1 + \alpha)*(1 + \beta)*(1 + \alpha\beta)^n = \left(2^{n+1} + (1 + \alpha\beta)^n\right)
$$

Finally, for the original program: $F_P = (F_{P'})^n = \left(2^{n+1} + (1 + \alpha\beta)^n\right)^n$. Note that this is also in closed form.

## 5 Experiments

The algorithm that we described is based on computing the generating functions of counting programs to perform lifted inference, which approaches the problem from a completely different angle than existing techniques. Due to this novelty, we can solve MLNs that are intractable for other existing lifted algorithms such as first-order variable elimination (FOVE) [5, 6, 7]. Specifically, we demonstrate with our experiments that on some MLNs we indeed outperform FOVE by orders of magnitude.

We ran our algorithm on the MLN given in Table 1. The set of features used in this MLN fall into the class of counting programs having a pseudo-polynomial generating function. This is the most general class of features our approach covers, and here our algorithm does not give any guarantees as evidence increases. The evidence in our experiments is randomly generated for the two tables `Asthma` and `Smokes`. In our experiments we study the influence of two factors on the runtime:

**Size of Domain:** Identifying tractable features is particularly important for inference in first order models, because (i) grounding can produce very big graphical models and (ii) the treewidth of these models could be very high. As the size of domain increases, our approach should scale better than the existing techniques which can't do lifted inference on this MLN. All the predicates in this MLN are only defined on one domain, that of *persons*.

**Evidence:** Since this MLN falls into the class of features for which we give no guarantees as evidence increases, we want to study the behavior of our algorithm in the presence of increasingly more evidence.

Fig. 5 displays the execution time of our CP algorithm vs the FOVE approach for domain sizes varying from 5 to 100, at the presence of 30% evidence. All results display average runtimes over 15 repetitions with the same parameter settings. FOVE cannot do lifted inference on this MLN and resorts to grounding. Thus, it could only execute up to the domain size of 18; after that it consistently ran out of memory. The figure also displays the extrapolated data points for FOVE's behavior in larger domain sizes, and shows its runtime growing exponentially. Our approach on the other hand dominates FOVE by orders of magnitude for those small domains, and finishes within seconds even for domains of size 100. Note that the complexity of our algorithm for this MLN is quadratic. Hence it looks linear on the log-scale.

Fig. 5 demonstrates the behavior of the algorithms as the amount of evidence is increased from 0 to 100%. We chose a domain size of 13 to run FOVE, since it couldn't terminate for higher domain sizes. The figure displays the runtime of our algorithm for domain sizes of 13 and 100. Although for this class of features we do not give guarantees on the running time for large evidence, our algorithm still performs well as the evidence increases. In fact after a point the algorithm gets faster. This is because the main time-consuming rule used in this MLN is **R4**. **R4** chooses a singleton atom in the last feature, say `Asthma`, and eliminates it. This involves time complexity proportional to the domain of the atom and the running time of the smaller MLN obtained after removing that atom. As evidence increases, the atom corresponding to `Asthma` may be split into many smaller predicates; but the domain size of each predicate also keeps getting smaller. In particular with 100% evidence, the domain is just 1 and therefore **R6** takes constant time!

## 6 Conclusion and Future Work

We have presented a novel approach to lifted inference that uses the theory of generating functions to do efficient inference. We also give guarantees on the theoretical complexity of our approach. This is the first work that tries to address the complexity of lifted inference in terms of only the features (formulas). This is beneficial because using a set of tractable features ensures that inference is always efficient and hence it will scale to large domains.

Several avenues remain for future work. For instance, a feature such as transitive closure ( e.g., `Friends(X,Y)` $\wedge$ `Friends(Y,Z)` $\Rightarrow$ `Friends(X,Z)`), which occurs quite often in many real world applications, is intractable for our algorithm. In future, we would like to address the complexity of such features by characterizing the completeness of our approach. Another avenue for future work is extending other lifted inference approaches [5, 7] with rules that we have developed in this paper. Unlike our algorithm, the aforementioned algorithms are complete. Namely, when lifted inference is not possible, they ground the domain and resort to propositional inference. But even in those cases, just running a propositional algorithm that does not exploit symmetry is not very efficient. In particular, ground networks generated by logical formulas have some repetition in their structure that is difficult to capture after grounding. Take for example `R(X,Y)` $\wedge$ `S(Z,Y)`. This feature is in PTIME by our algorithm, but if we create a ground markov network by grounding this feature then it can have unbounded treewidth (as big as the domain itself). We think our approach can provide an insight about how to best construct a graphical model from the groundings of a logical formula. This is also another interesting piece of future work that our algorithm motivates.

## References

[1] Lise Getoor and Ben Taskar. *Introduction to Statistical Relational Learning*. The MIT Press, 2007.

[2] Pedro Domingos and Daniel Lowd. *Markov Logic: An Interface Layer for Artificial Intelligence*. Morgan and Claypool, 2009.

[3] Matthew Richardson and Pedro Domingos. Markov logic networks. In *Machine Learning*, page 2006, 2006.

[4] David Poole. First-order probabilistic inference. In *IJCAI'03: Proceedings of the 18th international joint conference on Artificial intelligence*, pages 985–991, San Francisco, CA, USA, 2003. Morgan Kaufmann Publishers Inc.

[5] Rodrigo De Salvo Braz, Eyal Amir, and Dan Roth. Lifted first-order probabilistic inference. In *IJCAI'05: Proceedings of the 19th international joint conference on Artificial intelligence*, pages 1319–1325, San Francisco, CA, USA, 2005. Morgan Kaufmann Publishers Inc.

[6] Brian Milch, Luke S. Zettlemoyer, Kristian Kersting, Michael Haimes, and Leslie Pack Kaelbling. Lifted probabilistic inference with counting formulas. In *AAAI'08: Proceedings of the 23rd national conference on Artificial intelligence*, pages 1062–1068. AAAI Press, 2008.

[7] K. S. Ng, J. W. Lloyd, and W. T. Uther. Probabilistic modelling, inference and learning using logical theories. *Annals of Mathematics and Artificial Intelligence*, 54(1-3):159–205, 2008.

[8] Nevin Zhang and David Poole. A simple approach to bayesian network computations. In *Proceedings of the Tenth Canadian Conference on Artificial Intelligence*, pages 171–178, 1994.

[9] R. Dechter. Bucket elimination: A unifying framework for reasoning. *Artificial Intelligence*, 113:41–85, 1999.

[10] Parag Singla and Pedro Domingos. Lifted first-order belief propagation. In *AAAI'08: Proceedings of the 23rd national conference on Artificial intelligence*, pages 1094–1099. AAAI Press, 2008.

[11] J. Pearl. *Probabilistic Reasoning in Intelligent Systems*. Morgan Kaufmann, 1988.

[12] Kevin P. Murphy, Yair Weiss, and Michael I. Jordan. Loopy belief propagation for approximate inference: An empirical study. In *In Proceedings of the Fifteenth Conference on Uncertainty in Artificial Intelligence (UAI)*, pages 467–475, 1999.

[13] Nilesh Dalvi and Dan Suciu. Management of probabilistic data: foundations and challenges. In *PODS*, pages 1–12, New York, NY, USA, 2007. ACM Press.

[14] Karl Schnaitter Nilesh Dalvi and Dan Suciu. Computing query probability with incidence algebras. In *PODS*, 2007.

